# Sharing Clusters Among Related Groups: Hierarchical Dirichlet Processes

**Yee Whye Teh**[(1)]**, Michael I. Jordan**[(1,2)]**, Matthew J. Beal**[(3)] **and David M. Blei**[(1)]

[(1)]Computer Science Div., [(2)]Dept. of Statistics
University of California at Berkeley
Berkeley CA 94720, USA
{ywteh,jordan,blei}@cs.berkeley.edu

[(3)]Dept. of Computer Science
University of Toronto
Toronto M5S 3G4, Canada
beal@cs.toronto.edu

## Abstract

We propose the hierarchical Dirichlet process (HDP), a nonparametric Bayesian model for clustering problems involving multiple groups of data. Each group of data is modeled with a mixture, with the number of components being open-ended and inferred automatically by the model. Further, components can be shared across groups, allowing dependencies across groups to be modeled effectively as well as conferring generalization to new groups. Such grouped clustering problems occur often in practice, e.g. in the problem of topic discovery in document corpora. We report experimental results on three text corpora showing the effective and superior performance of the HDP over previous models.

## 1 Introduction

One of the most significant conceptual and practical tools in the Bayesian paradigm is the notion of a *hierarchical model*. Building on the notion that a parameter is a random variable, hierarchical models have applications to a variety of forms of grouped or relational data and to general problems involving "multi-task learning" or "learning to learn." A simple and classical example is the Gaussian means problem, in which a grand mean $\mu_0$ is drawn from some distribution, a set of $K$ means are then drawn independently from a Gaussian with mean $\mu_0$, and data are subsequently drawn independently from $K$ Gaussian distributions with these means. The posterior distribution based on these data couples the means, such that posterior estimates of the means are shrunk towards each other. The estimates "share statistical strength," a notion that can be made precise within both the Bayesian and the frequentist paradigms.

Here we consider the application of hierarchical Bayesian ideas to a problem in "multi-task learning" in which the "tasks" are clustering problems, and our goal is to share clusters among multiple, related clustering problems. We are motivated by the task of discovering topics in document corpora [1]. A topic (i.e., a cluster) is a distribution across words while documents are viewed as distributions across topics. We want to discover topics that are common across multiple documents in the same corpus, as well as across multiple corpora.

Our work is based on a tool from nonparametric Bayesian analysis known as the *Dirichlet process* (DP) mixture model [2, 3]. Skirting technical definitions for now, "nonparametric"

can be understood simply as implying that the number of clusters is open-ended. Indeed, at each step of generating data points, a DP mixture model can either assign the data point to a previously-generated cluster or can start a new cluster. The number of clusters is a random variable whose mean grows at rate logarithmic in the number of data points.

Extending the DP mixture model framework to the setting of multiple related clustering problems, we will be able to make the (realistic) assumption that we do not know the number of clusters a priori in any of the problems, nor do we know how clusters should be shared among the problems.

When generating a new cluster, a DP mixture model selects the parameters for the cluster (e.g., in the case of Gaussian mixtures, the mean and covariance matrix) from a distribution $G_0$—the *base distribution*. So as to allow any possible parameter value, the distribution $G_0$ is often assumed to be a smooth distribution (i.e., non-atomic). Unfortunately, if we now wish to extend DP mixtures to groups of clustering problems, the assumption that $G_0$ is smooth conflicts with the goal of sharing clusters among groups. That is, even if each group shares the same underlying base distribution $G_0$, the smoothness of $G_0$ implies that they will generate distinct cluster parameters (with probability one). We will show that this problem can be resolved by taking a hierarchical Bayesian approach. We present a notion of a *hierarchical Dirichlet process* (HDP) in which the base distribution $G_0$ for a set of DPs is itself a draw from a DP. This turns out to provide an elegant and simple solution to the problem of sharing clusters among multiple clustering problems.

The paper is organized as follows. In Section 2, we provide the basic technical definition of DPs and discuss related representations involving stick-breaking processes and Chinese restaurant processes. Section 3 then introduces the HDP, motivated by the requirement of a more powerful formalism for the grouped data setting. As for the DP, we present analogous stick-breaking and Chinese restaurant representations for the HDP. We present empirical results on a number of text corpora in Section 5, demonstrating various aspects of the HDP including its nonparametric nature, hierarchical nature, and the ease with which the framework can be applied to other realms such as hidden Markov models.

## 2   Dirichlet Processes

The Dirichlet process (DP) and the DP mixture model are mainstays of nonparametric Bayesian statistics (see, e.g., [3]). They have also begun to be seen in applications in machine learning (e.g., [7, 8, 9]). In this section we give a brief overview with an eye towards generalization to HDPs. We begin with the definition of DPs [4]. Let $(\Theta, \mathcal{B})$ be a measurable space, with $G_0$ a probability measure on the space, and let $\alpha_0$ be a positive real number. A *Dirichlet process* is the distribution of a random probability measure $G$ over $(\Theta, \mathcal{B})$ such that, for any finite partition $(A_1, \ldots, A_r)$ of $\Theta$, the random vector $(G(A_1), \ldots, G(A_r))$ is distributed as a finite-dimensional Dirichlet distribution:

$$(G(A_1), \ldots, G(A_r)) \sim \mathrm{Dir}\big(\alpha_0 G_0(A_1), \ldots, \alpha_0 G_0(A_r)\big) . \tag{1}$$

We write $G \sim \mathrm{DP}(\alpha_0, G_0)$ if $G$ is a random probability measure distributed according to a DP. We call $G_0$ the base measure of $G$, and $\alpha_0$ the concentration parameter.

The DP can be used in the mixture model setting in the following way. Consider a set of data, $\mathbf{x} = (x_1, \ldots, x_n)$, assumed exchangeable. Given a draw $G \sim \mathrm{DP}(\alpha_0, G_0)$, independently draw $n$ *latent factors* from $G$: $\phi_i \sim G$. Then, for each $i = 1, \ldots, n$, draw $x_i \sim F(\phi_i)$, for a distribution $F$. This setup is referred to as a *DP mixture model*.

If the factors $\phi_i$ were all distinct, then this setup would yield an (uninteresting) mixture model with $n$ components. In fact, the DP exhibits an important *clustering property*, such that the draws $\phi_i$ are generally *not* distinct. Rather, the number of distinct values grows as $O(\log n)$, and it is this that defines the random number of mixture components.

There are several perspectives on the DP that help to understand this clustering property. In this paper we will refer to two: the *Chinese restaurant process* (CRP), and the *stick-breaking process*. The CRP is a distribution on partitions that directly captures the clustering of draws from a DP via a metaphor in which customers share tables in a Chinese restaurant [5]. As we will see in Section 4, the CRP refers to properties of the joint distribution of the factors $\{\phi_i\}$. The stick-breaking process, on the other hand, refers to properties of $G$, and directly reveals its discrete nature [6]. For $k = 1, 2 \ldots$, let:

$$\theta_k \sim G_0 \qquad \beta'_k \sim \text{Beta}(1, \alpha_0) \qquad \beta_k = \beta'_k \prod_{l=1}^{k-1}(1 - \beta'_k). \qquad (2)$$

Then with probability one the random measure defined by $G = \sum_{k=1}^{\infty} \beta_k \delta_{\theta_k}$ is a sample from $\text{DP}(\alpha_0, G_0)$. The construction for $\beta_1, \beta_2, \ldots$ in (2) can be understood as taking a stick of unit length, and repeatedly breaking off segments of length $\beta_k$. The stick-breaking construction shows that DP mixture models can be viewed as mixture models with a countably infinite number of components. To see this, identify each $\theta_k$ as the parameter of the $k^{\text{th}}$ mixture component, with mixing proportion given by $\beta_k$.

## 3 Hierarchical Dirichlet Processes

We will introduce the *hierarchical Dirichlet process* (HDP) in this section. First we describe the general setting in which the HDP is most useful—that of *grouped data*. We assume that we have $J$ groups of data, each consisting of $n_j$ data points $(x_{j1}, \ldots, x_{jn_j})$. We assume that the data points in each group are exchangeable, and are to be modeled with a mixture model. While each mixture model has mixing proportions specific to the group, we require that the different groups share the same set of mixture components. The idea is that while different groups have different characteristics given by a different combination of mixing proportions, using the same set of mixture components allows statistical strength to be shared across groups, and allows generalization to new groups.

The HDP is a nonparametric prior which allows the mixture models to share components. It is a distribution over a set of random probability measures over $(\Theta, \mathcal{B})$: one probability measure $G_j$ for each group $j$, and a global probability measure $G_0$. The global measure $G_0$ is distributed as $\text{DP}(\gamma, H)$, with $H$ the base measure and $\gamma$ the concentration parameter, while each $G_j$ is conditionally independent given $G_0$, with distribution $G_j \sim \text{DP}(\alpha_0, G_0)$. To complete the description of the HDP mixture model, we associate each $x_{ji}$ with a factor $\phi_{ji}$, with distributions given by $F(\phi_{ji})$ and $G_j$ respectively. The overall model is given in Figure 1 left, with conditional distributions:

$$G_0 \,|\, \gamma, H \sim \text{DP}(\gamma, H) \qquad\qquad G_j \,|\, \alpha, G_0 \sim \text{DP}(\alpha_0, G_0) \qquad (3)$$
$$\phi_{ji} \,|\, G_j \sim G_j \qquad\qquad\qquad\qquad x_{ji} \,|\, \phi_{ji} \sim F(\phi_{ji}) \,. \qquad (4)$$

The stick-breaking construction (2) shows that a draw of $G_0$ can be expressed as a weighted sum of point masses: $G_0 = \sum_{k=1}^{\infty} \beta_k \delta_{\theta_k}$. This fact that $G_0$ is atomic plays an important role in ensuring that mixture components are shared across different groups. Since $G_0$ is the base distribution for the individual $G_j$'s, (2) again shows that the atoms of the individual $G_j$ are samples from $G_0$. In particular, since $G_0$ places non-zero mass only on the atoms $\boldsymbol{\theta} = (\theta_k)_{k=1}^{\infty}$, the atoms of $G_j$ must also come from $\boldsymbol{\theta}$, hence we may write:

$$G_0 = \sum_{k=1}^{\infty} \beta_k \delta_{\theta_k} \qquad\qquad G_j = \sum_{k=1}^{\infty} \pi_{jk} \delta_{\theta_k} \,. \qquad (5)$$

Identifying $\theta_k$ as the parameters of the $k^{\text{th}}$ mixture component, we see that each submodel corresponding to distinct groups share the same set of mixture components, but have differing mixing proportions, $\boldsymbol{\pi}_j = (\pi_{jk})_{k=1}^{\infty}$.

Finally, it is useful to explicitly describe the relationships between the mixing proportions $\boldsymbol{\beta}$ and $(\boldsymbol{\pi}_j)_{j=1}^{J}$. Details are provided in [10]. Note that the weights $\boldsymbol{\pi}_j$ are conditionally independent given $\boldsymbol{\beta}$ since each $G_j$ is independent given $G_0$. Applying (1) to finite partitions

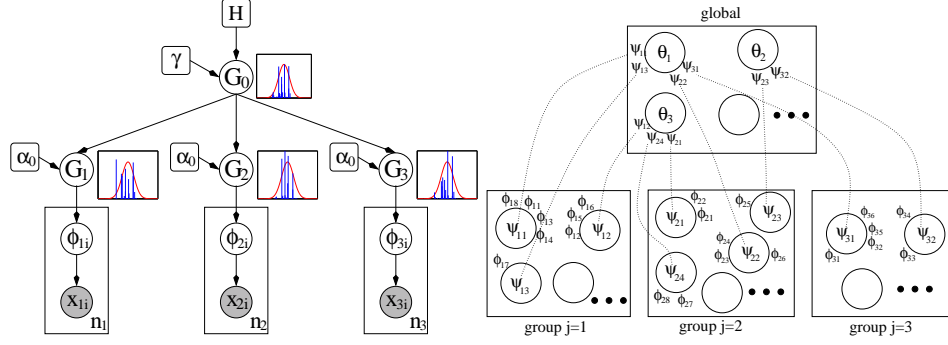

Figure 1: Left: graphical model of an example HDP mixture model with 3 groups. Corresponding to each DP node we also plot a sample draw from the DP using the stick-breaking construction. Right: an instantiation of the CRF representation for the 3 group HDP. Each of the 3 restaurants has customers sitting around tables, and each table is served a dish (which corresponds to customers in the Chinese restaurant for the global DP).

of $\theta$, we get $\pi_j \sim \mathrm{DP}(\alpha_0, \beta)$, where we interpret $\beta$ and $\pi_j$ as probability measures over the positive integers. Hence $\beta$ is simply the putative mixing proportion over the groups. We may in fact obtain an explicit stick-breaking construction for the $\pi_j$'s as well. Applying (1) to partitions $(\{1, \ldots, k-1\}, \{k\}, \{k+1, \ldots\})$ of positive integers, we have:

$$\pi'_{jk} \sim \mathrm{Beta}\left(\alpha_0 \beta_k, \alpha_0 \left(1 - \sum_{l=1}^{k} \beta_l\right)\right) \qquad \pi_{jk} = \pi'_{jk} \prod_{l=1}^{k-1} (1 - \pi'_{jl}) . \qquad (6)$$

## 4 The Chinese Restaurant Franchise

We describe an alternative view of the HDP based directly upon the distribution a HDP induces on the samples $\phi_{ji}$, where we marginalize out $G_0$ and $G_j$'s. This view directly leads to an efficient Gibbs sampler for HDP mixture models, which is detailed in the appendix. Consider, for one group $j$, the distribution of $\phi_{j1}, \ldots, \phi_{jn_j}$ as we marginalize out $G_j$. Recall that since $G_j \sim \mathrm{DP}(\alpha_0, G_0)$ we can describe this distribution by describing how to generate $\phi_{j1}, \ldots, \phi_{jn_j}$ using the CRP. Imagine $n_j$ customers (each corresponds to a $\phi_{ji}$) at a Chinese restaurant with an unbounded number of tables. The first customer sits at the first table. A subsequent customer sits at an occupied table with probability proportional to the number of customers already there, or at the next unoccupied table with probability proportional to $\alpha_0$. Suppose customer $i$ sat at table $t_{ji}$. The conditional distributions are:

$$t_{ji} \mid t_{j1}, \ldots, t_{ji-1}, \alpha_0 \sim \sum_t \frac{n_{jt}}{\sum_{t'} n_{jt'} + \alpha_0} \delta_t + \frac{\alpha_0}{\sum_{t'} n_{jt'} + \alpha_0} \delta_{t^{new}} , \qquad (7)$$

where $n_{jt}$ is the number of customers currently at table $t$. Once all customers have sat down the seating plan corresponds to a partition of $\phi_{j1}, \ldots, \phi_{jn_j}$. This is an exchangeable process in that the probability of a partition does not depend on the order in which customers sit down. Now we associate with table $t$ a draw $\psi_{jt}$ from $G_0$, and assign $\phi_{ji} = \psi_{jt_{ji}}$.

Performing this process independently for each group $j$, we have now integrated out all the $G_j$'s, and have an assignment of each $\phi_{ji}$ to a sample $\psi_{jt_{ji}}$ from $G_0$, with the partition structures given by CRPs. Notice now that all $\psi_{jt}$'s are simply i.i.d. draws from $G_0$, which is again distributed according to $\mathrm{DP}(\gamma, H)$, so we may apply the same CRP partitioning process to the $\psi_{jt}$'s. Let the customer associated with $\psi_{jt}$ sit at table $k_{jt}$. We have:

$$k_{jt} \mid k_{11}, \ldots, k_{1n_1}, k_{21}, \ldots, k_{jt-1}, \gamma \sim \sum_k \frac{m_k}{\sum_{k'} m_{jk'} + \gamma} \delta_k + \frac{\gamma}{\sum_{k'} m_{k'} + \alpha_0} \delta_{k^{new}} . \qquad (8)$$

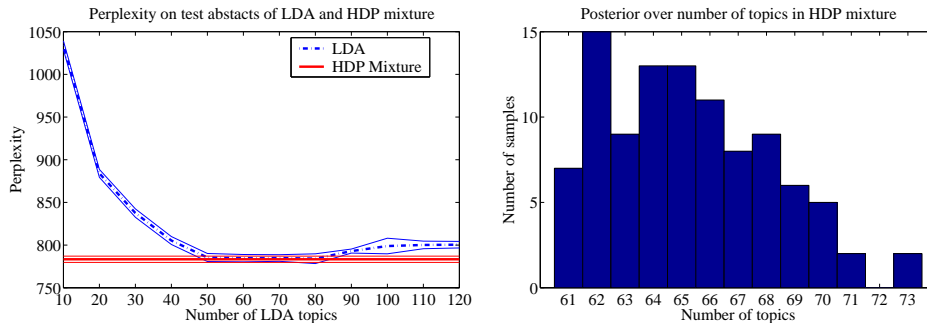

Figure 2: Left: comparison of LDA and HDP mixture. Results are averaged over 10 runs, with error bars being 1 standard error. Right: histogram of the number of topics the HDP mixture used over 100 posterior samples.

Finally we associate with table $k$ a draw $\theta_k$ from $H$ and assign $\psi_{jt} = \theta_{k_{jt}}$. This completes the generative process for the $\phi_{ji}$'s, where we marginalize out $G_0$ and $G_j$'s. We call this generative process the *Chinese restaurant franchise* (CRF). The metaphor is as follows: we have $J$ restaurants, each with $n_j$ customers ($\phi_{ji}$'s), who sit at tables ($\psi_{jt}$'s). Now each table is served a dish ($\theta_k$'s) from a menu common to all restaurants. The customers are sociable, preferring large tables with many customers present, and also prefer popular dishes.

## 5   Experiments

We describe 3 experiments in this section to highlight the various aspects of the HDP: its nonparametric nature; its hierarchical nature; and the ease with which we can apply the framework to other models, specifically the HMM.

**Nematode biology abstracts.** To demonstrate the strength of the nonparametric approach as exemplified by the HDP mixture, we compared it against *latent Dirichlet allocation* (LDA), which is a parametric model similar in structure to the HDP [1]. In particular, we applied both models to a corpus of nematode biology abstracts[1], evaluating the perplexity of both models on held out abstracts. Here abstracts correspond to groups, words correspond to observations, and topics correspond to mixture components, and exchangeability correspond to the typical bag-of-words assumption. In order to study specifically the nonparametric nature of the HDP, we used the same experimental setup for both models[2], except that in LDA we had to vary the number of topics used between 10 and 120, while the HDP obtained posterior samples over this automatically.

The results are shown in Figure 2. LDA performs best using between 50 and 80 topics, while the HDP performed just as well as these. Further, the posterior over the number of topics used by HDP is consistent with this range. Notice however that the HDP infers the number of topics automatically, while LDA requires some method of model selection.

**NIPS sections.** We applied HDP mixture models to a dataset of NIPS 1-12 papers organized into sections[3]. To highlight the transfer of learning achievable with the HDP, we

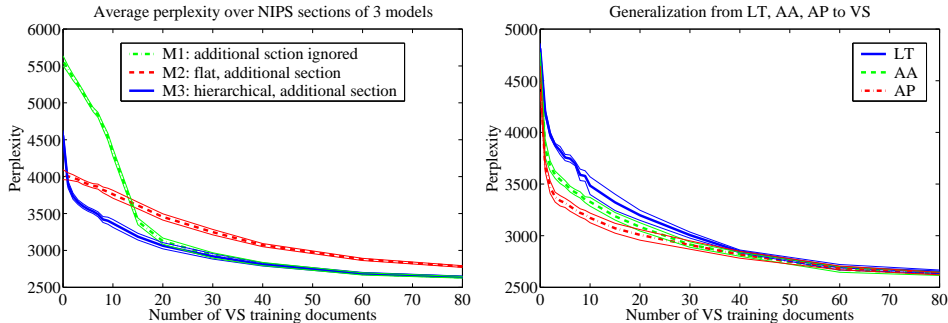

Figure 3: Left: perplexity of test VS documents given training documents from VS and another section for 3 different models. Curves shown are averaged over the other sections and 5 runs. Right: perplexity of test VS documents given LT, AA and AP documents respectively, using M3, averaged over 5 runs. In both, the error bars are 1 standard error.

show improvements to the modeling of a section when the model is also given documents from another section. Our test section is always the VS (vision sciences) section, while the additional section is varied across the other eight. The training set always consist of 80 documents from the other section (so that larger sections like AA (algorithms and architecures) do not get an unfair advantage), plus between 0 and 80 documents from VS. There are 47 test documents, which are held fixed as we vary over the other section and the number $N$ of training VS documents. We compared 3 different models for this task. The first model (M1) simply ignores documents from the additional section, and uses a HDP to model the VS documents. It serves as a baseline. The second model (M2) uses a HDP mixture model, with one group per document, but lumping together training documents from both sections. The third model (M3) takes a hierarchical approach and models each section separately using a HDP mixture model, and places another DP prior over the common base distributions for both submodels[4].

As we see in Figure 3 left, the more hierarchical approach of M3 performs best, with perplexity decreasing drastically with modest values of $N$, while M1 does worst for small $N$. However with increasing $N$, M1 improves until it is competitive with M3 but M2 does worst. This is because M2 lumps all the documents together, so is not able to differentiate between the sections, as a result the influence of documents from the other section is unduly strong. This result confirms that the hierarchical approach to the transfer-of-learning problem is a useful one, as it allows useful information to be transfered to a new task (here the modeling of a new section), without the data from the previous tasks overwhelming those in the new task.

We also looked at the performance of the M3 model on VS documents given specific other sections. This is shown in Figure 3 right. As expected, the performance is worst given LT (learning theory), and improves as we move to AA and AP (applications). In Table 1 we show the topics pertinent to VS discovered by the M3 model. First we trained the model on all documents from the other section. Then, keeping the assignments of words to topics fixed in the other section, we introduced VS documents and the model decides to reuse some topics from the other section, as well as create new ones. The topics reused by VS documents confirm to our expectations of the overlap between VS and other sections.

as words occurring more than 4000 or less than 50 times in the documents. As sections differ over the years, we assigned by hand the various sections to one of 9 prototypical sections: CS, NS, LT, AA, IM, SP, VS, AP and CN.

[4]Though we have only described the 2 layer HDP the 3 layer extension is straightforward. In fact on our website http://www.cs.berkeley.edu/~ywteh/research/npbayes we have an implementation of the general case where DPs are coupled hierarchically in a tree-structured model.

| CS | NS | LT | AA | IM | SP | AP | CN |
|---|---|---|---|---|---|---|---|
| task representation pattern processing trained representations three process unit patterns | cells cell activity response neuron visual patterns pattern single fig | signal layer gaussian cells fig nonlinearity nonlinear rate eq cell | algorithms test approach methods based point problems form large paper | processing pattern approach architecture single shows simple based large control | visual images video language image pixel acoustic delta lowpass flow | approach based trained test layer features table classification rate paper | ii tree pomdp observable strategy class stochastic history strategies density |
| examples concept similarity bayesian hypotheses generalization numbers positive classes hypothesis | visual cells cortical orientation receptive contrast spatial cortex stimulus tuning | large examples form point see parameter consider random small optimal | distance tangent image images transformation transformations pattern vectors convolution simard | motion visual velocity flow target chip eye smooth direction optical | signals separation signal sources source matrix blind mixing gradient eq | image images face similarity pixel visual database matching facial examples | policy optimal reinforcement control action states actions step problems goal |

Table 1: Topics shared between VS and the other sections. Shown are the two topics with most numbers of VS words, but also with significant numbers of words from the other section.

**Alice in Wonderland.** The *infinite hidden Markov model* (iHMM) is a nonparametric model for sequential data where the number of hidden states is open-ended and inferred from data [11]. In [10] we show that the HDP framework can be applied to obtain a cleaner formulation of the iHMM, providing effective new inference algorithms and potentially hierarchical extensions. In fact the original iHMM paper [11] served as inspiration for this work and first coined the term "hierarchical Dirichlet processes"—though their model is not hierarchical in the Bayesian sense, involving priors upon priors, but is rather a set of coupled urn models similar to the CRF. Here we report experimental comparisons of the iHMM against other approaches on sentences taken from Lewis Carroll's *Alice's Adventures in Wonderland*.

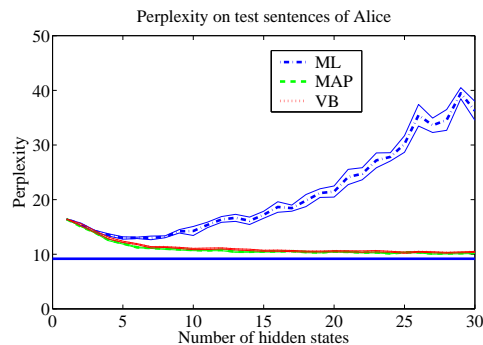

Figure 4: Comparing iHMM (horizontal line) versus ML, MAP and VB trained HMMs. Error bars are 1 standard error (those for iHMM too small to see).

ML, MAP, and variational Bayesian (VB) [12] models with numbers of states ranging from 1 to 30 were trained multiple times on 20 sentences of average length 51 symbols (27 distinct symbols, consisting of 26 letters and ' '), and tested on 40 sequences of average length 100. Figure 4 shows the perplexity of test sentences. For VB, the predictive probability is intractable to compute, so the modal setting of parameters was used. Both MAP and VB models were given optimal settings of the hyperparameters found in the iHMM. We see that the iHMM has a lower perplexity than every model size for ML, MAP, and VB, and obtains this with one countably infinite model.

## 6 Discussion

We have described the hierarchical Dirichlet process, a hierarchical, nonparametric model for clustering problems involving multiple groups of data. HDP mixture models are able to automatically determine the appropriate number of mixture components needed, and exhibit sharing of statistical strength across groups by having components shared across groups. We have described the HDP as a distribution over distributions, using both the stick-breaking construction and the Chinese restaurant franchise. In [10] we also describe a fourth perspective based on the infinite limit of finite mixture models, and give detail for

how the HDP can be applied to the iHMM. Direct extensions of the model include use of nonparametric priors other than the DP, building higher level hierarchies as in our NIPS experiment, as well as hierarchical extensions to the iHMM.

**Appendix: Gibbs Sampling in the CRF**

The CRF is defined by the variables $\boldsymbol{t} = (t_{ji})$, $\boldsymbol{k} = (k_{jt})$, and $\boldsymbol{\theta} = (\theta_k)$. We describe an inference procedure for the HDP mixture model based on Gibbs sampling $\boldsymbol{t}, \boldsymbol{k}$ and $\boldsymbol{\theta}$ given data items $\mathbf{x}$. For the full derivation see [10]. Let $f(\cdot|\theta)$ and $h$ be the density functions for $F(\theta)$ and $H$ respectively, $n_{jt}^{-i}$ be the number of $t_{ji'}$'s equal to $t$ except $t_{ji}$, and $m_k^{-jt}$ be the number of $k_{j't'}$'s equal to $k$ except $k_{jt}$. The conditional probability for $t_{ji}$ given the other variables is proportional to the product of a prior and likelihood term. The prior term is given by (7) where, by exchangeability, we can take $t_{ji}$ to be the last one assigned. The likelihood is given by $f(x_{ji}|\theta_{k_{jt}})$ where for $t = t^{\text{new}}$ we may sample $k_{jt^{\text{new}}}$ using (8), and $\theta_{k^{\text{new}}} \sim H$. The distribution is then:

$$p(t_{ji} = t \,|\, \boldsymbol{t} \backslash t_{ji}, \boldsymbol{k}, \boldsymbol{\theta}, \mathbf{x}) \propto \begin{cases} \alpha_0 f(x_{ji}|\theta_{k_{jt}}) & \text{if } t = t^{\text{new}} \\ n_{jt}^{-i} f(x_{ji}|\theta_{k_{jt}}) & \text{if } t \text{ currently used.} \end{cases} \tag{9}$$

Similarly the conditional distributions for $k_{jt}$ and $\theta_k$ are:

$$p(k_{jt} = k \,|\, \boldsymbol{t}, \boldsymbol{k} \backslash k_{jt}, \boldsymbol{\theta}, \mathbf{x}) \propto \begin{cases} \gamma \prod_{i:t_{ji}=t} f(x_{ji}|\theta_k) & \text{if } k = k^{\text{new}} \\ m_k^{-t} \prod_{i:t_{ji}=t} f(x_{ji}|\theta_k) & \text{if } k \text{ currently used.} \end{cases} \tag{10}$$

$$p(\theta_k \,|\, \boldsymbol{t}, \boldsymbol{k}, \boldsymbol{\theta} \backslash \theta_k, \mathbf{x}) \propto h(\theta_k) \prod_{ji:k_{jt_{ji}}=k} f(x_{ji}|\theta_k) \tag{11}$$

where $\theta_{k^{\text{new}}} \sim H$. If $H$ is conjugate to $F(\cdot)$ we have the option of integrating out $\boldsymbol{\theta}$.

**References**

[1] D.M. Blei, A.Y. Ng, and M.I. Jordan. Latent Dirichlet allocation. *JMLR*, 3:993–1022, 2003.

[2] M.D. Escobar and M. West. Bayesian density estimation and inference using mixtures. *Journal of the American Statistical Association*, 90:577–588, 1995.

[3] S.N. MacEachern and P. Müller. Estimating mixture of Dirichlet process models. *Journal of Computational and Graphical Statistics*, 7:223–238, 1998.

[4] T.S. Ferguson. A Bayesian analysis of some nonparametric problems. *Annals of Statistics*, 1(2):209–230, 1973.

[5] D. Aldous. Exchangeability and related topics. In *École d'été de probabilités de Saint-Flour XIII–1983*, pages 1–198. Springer, Berlin, 1985.

[6] J. Sethuraman. A constructive definition of Dirichlet priors. *Statistica Sinica*, 4:639–650, 1994.

[7] R.M. Neal. Markov chain sampling methods for Dirichlet process mixture models. *Journal of Computational and Graphical Statistics*, 9:249–265, 2000.

[8] C.E. Rasmussen. The infinite Gaussian mixture model. In *NIPS*, volume 12, 2000.

[9] D.M. Blei, T.L. Griffiths, M.I. Jordan, and J.B. Tenenbaum. Hierarchical topic models and the nested Chinese restaurant process. NIPS, 2004.

[10] Y.W. Teh, M.I. Jordan, M.J. Beal, and D.M. Blei. Hierarchical dirichlet processes. Technical Report 653, Department of Statistics, University of California at Berkeley, 2004.

[11] M.J. Beal, Z. Ghahramani, and C.E. Rasmussen. The infinite hidden Markov model. In *NIPS*, volume 14, 2002.

[12] M.J. Beal. *Variational Algorithms for Approximate Bayesian Inference*. PhD thesis, Gatsby Unit, University College London, 2004.

## Footnotes

[1]Available at http://elegans.swmed.edu/wli/cgcbib. There are 5838 abstracts in total. After removing standard stop words and words appearing less than 10 times, we are left with 476441 words in total and a vocabulary size of 5699.

[2]In both models, we used a symmetric Dirichlet distribution with weights of $0.5$ for the prior $H$ over topic distributions, while the concentration parameters are integrated out using a vague gamma prior. Gibbs sampling using the CRF is used, while the concentration parameters are sampled using a method described in [10]. This also applies to the NIPS sections experiment on next page.

[3]To ensure we are dealing with informative words in the documents, we culled stop words as well
